# On Stochastic Complexity and Admissible Models for Neural Network Classifiers

**Padhraic Smyth**
Communications Systems Research
Jet Propulsion Laboratory
California Institute of Technology
Pasadena, CA 91109

## Abstract

Given some training data how should we choose a particular network classifier from a family of networks of different complexities? In this paper we discuss how the application of stochastic complexity theory to classifier design problems can provide some insights into this problem. In particular we introduce the notion of *admissible* models whereby the complexity of models under consideration is affected by (among other factors) the class entropy, the amount of training data, and our prior belief. In particular we discuss the implications of these results with respect to neural architectures and demonstrate the approach on real data from a medical diagnosis task.

## 1  Introduction and Motivation

In this paper we examine in a general sense the application of Minimum Description Length (MDL) techniques to the problem of selecting a good classifier from a large set of candidate models or hypotheses. Pattern recognition algorithms differ from more conventional statistical modeling techniques in the sense that they typically choose from a very large number of candidate models to describe the available data. Hence, the problem of searching through this set of candidate models is frequently a formidable one, often approached in practice by the use of greedy algorithms. In this context, techniques which allow us to eliminate portions of the hypothesis space are of considerable interest. We will show in this paper that it is possible to use the intrinsic structure of the MDL formalism to eliminate large numbers of candidate models given only minimal information about the data. Our results depend on the

very simple notion that models which are obviously too complex for the problem (e.g., models whose complexity exceeds that of the data itself) can be discarded from further consideration in the search for the most parsimonious model.

## 2 Background on Stochastic Complexity Theory

### 2.1 General Principles

Stochastic complexity prescribes a general theory of inductive inference from data, which, unlike more traditional inference techniques, takes into account the *complexity* of the proposed model in addition to the standard goodness-of-fit of the model to the data. For a detailed rationale the reader is referred to the work of Rissanen (1984) or Wallace and Freeman (1987) and the references therein. Note that the Minimum Description Length (MDL) technique (as Rissanen's approach has become known) is implicitly related to Maximum A Posteriori (MAP) Bayesian estimation techniques if cast in the appropriate framework.

### 2.2 Minimum Description Length and Stochastic Complexity

Following the notation of Barron and Cover (1991), we have $N$ data-points, described as a sequence of tuples of observations $\{x_i^1, \ldots, x_i^K, y_i\}, 1 \leq i \leq N$, to be referred to as $\{\underline{x_i}, y_i\}$ for short. The $x_i^k$ correspond to values taken on by the $K$ random variables $X^k$ (which may be continuous or discrete), while, for the purposes of this paper, the $y_i$ are elements of the finite alphabet of the discrete $m$-ary class variable $Y$. Let $\Gamma_N = \{M_1, \ldots, M_{|\Gamma_N|}\}$ be the family of candidate models under consideration. Note that by defining $\Gamma_N$ as a function of $N$, the number of data points, we allow the possibility of considering more complicated models as more data arrives. For each $M_j \in \Gamma_N$ let $C(M_j)$ be non-negative numbers such that

$$\sum_j 2^{-C(M_j)} \leq 1.$$

The $C(M_j)$ can be interpreted as the cost in bits of specifying model $M_j$ — in turn, $2^{-C(M_j)}$ is the prior probability assigned to model $M_j$ (suitably normalized). Let us use $\mathcal{C} = \{C(M_1), \ldots, C(M_{|\Gamma_N|})\}$ to refer to a particular coding scheme for $\Gamma_N$. Hence the total *description length* of the data plus a model $M_j$ is defined as

$$L(M_j, \{\underline{x_i}, y_i\}) = C(M_j) + \log\left(\frac{1}{p(\{y_i\}|M_j(\{\underline{x_i}\}))}\right)$$

i.e., we first describe the model and then the class data relative to the given model (as a function of $\{\underline{x_i}\}$, the feature data). The *stochastic complexity* of the data $\{\underline{x_i}, y_i\}$ relative to $\mathcal{C}$ and $\Gamma_N$ is the minimum description length

$$I(\{\underline{x_i}, y_i\}) = \min_{M_j \in \Gamma_N} \{L(M_j, \{\underline{x_i}, y_i\})\}.$$

The problem of finding the model of *shortest* description length is intractable in the general case — nonetheless the idea of finding the best model we can is well motivated, works well in practice and is far preferable to the alternative approach of ignoring the complexity issue entirely.

# 3  Admissible Stochastic Complexity Models

## 3.1  Definition of Admissibility

We will find it useful to define the notion of an *admissible* model for the classification problem: the set of admissible models $\Omega_N$ ($\subseteq \Gamma_N$) is defined as all models whose complexity is such that there exists no other model whose description length is known to be smaller. In other words we are saying that inadmissible models are those which have complexity in bits greater than any *known* description length — clearly they cannot be better than the best known model in terms of description length and can be eliminated from consideration. Hence, $\Omega_N$ is defined dynamically and is a function of how many description lengths we have already calculated in our search. Typically $\Gamma_N$ may be pre-defined, such as the class of all 3-layer feed-forward neural networks with particular activation functions. We would like to restrict our search for a good model to the set $\Omega_N \subseteq \Gamma_N$ as far as possible (since non-admissible models are of no practical use). In practice it may be difficult to determine the exact boundaries of $\Omega_N$, particularly when $|\Gamma_N|$ is large (with decision trees or neural networks for example). Note that the notion of admissibility described here is particularly useful when we seek a *minimal* description length, or equivalently a model of *maximal a posteriori* probability — in situations where one's goal is to average over a number of possible models (in a Bayesian manner) a modification of the admissibility criterion would be necessary.

## 3.2  Results for Admissible Models

Simple techniques for eliminating obvious non-admissible models are of interest: for the classification problem a necessary condition that a model $M_j$ be admissible is that

$$C(M_j) \leq N \cdot H(X) \leq N \log(m)$$

where $H(X)$ is the entropy of the $m$-ary class variable $X$. The obvious interpretation in words is that any admissible model must have complexity less than that of the data itself. It is easy to show in addition that the complexity of any admissible model is upper bounded by the parameters of the classification problem:

$$p(M_j) \geq 2^{-N.H(X)} \geq 2^{-N \log(m)}, \qquad \forall M_j \in \Omega_N.$$

Hence, the size of the space of admissible models can also be bounded:

$$|\Omega_N| \leq 2^{N \cdot H(X)} \leq 2^{N. \log(m)}.$$

Our approach suggests that for classification at least, once we know $N$ and the number of classes $m$, there are strict limitations on how many *admissible* models we can consider. Of course the theory does not state that considering a larger subset will necessarily result in a less optimal model being found, however, it is difficult to argue the case for including large numbers of models which are clearly too complex for the problem. At best, such an approach will lead to an inefficient search, whereas at worst a very poor model will be chosen perhaps as a result of the use of a poor coding scheme for the unnecessarily large hypothesis space.

## 3.3   Admissible Models and Bayes Risk

The notion of *minimal* compression (the minimum achievable goodness-of-fit) is intimately related in the classification problem to the minimal Bayes risk for the problem (Kovalevsky, 1980). Let $M_B$ be any model (not necessarily unique) which achieves the optimal Bayes risk (i.e., minimizes the classifier error) for the classification problem. In particular, $C(\{x_i\}|M_B(\{y_i\}))$ is not necessarily zero, indeed in most practical problems of interest it is non-zero, due to the ambiguity in the mapping from the feature space to the class variable. In addition, $M_B$ may not be defined in the set $\Gamma_N$, and hence, $M_B$ need not even be admissible. If, in the limit as $N \to \infty$, $M_B \notin \Gamma_\infty$ then there is a fundamental *approximation* error in the representation being used, i.e., the family of models under consideration is not flexible enough to optimally represent the mapping from $\{x_i\}$ to $\{y_i\}$. Smyth (1991) has shown how information about the Bayes error rate for the problem (if available) can be used to further tighten the bounds on admissibility.

# 4   Applying Minimum Description Length Principles to Neural Network Design

In principle the admissibility results can be applied to a variety of classifier design problems — applications to Markov model selection and decision tree design are described elsewhere (Smyth, 1991). In this paper we limit our attention to the problem of automatically selecting a feedforward multi-layer network architecture.

## 4.1   Calculation of the Goodness-of-Fit

As is clear from the preceding discussion, application of the MDL principle to classifier selection requires that the classifier produce a posterior probability estimate of the class labels. In the context of a network model this is not a problem provided the network is trained to provide such estimates. This requires a simple modification of the objective function to a log-likelihood function $-\sum_{i=1}^{N} \log(\hat{p}(y_i|x_i))$, where $y_i$ is the class label of the $i$th training datum and $\hat{p}()$ is the network's estimate of $p()$. This function has been proposed in the literature in the past under the guise of a cross-entropy measure (for the special case of binary classes) and more recently it has been derived from the more basic arguments of Minimum Mutual Information (MMI) (Bridle, 1990) and Maximum Likelihood (ML) Estimation (Gish, 1990). The cross-entropy function for network training is nothing more that the goodness-of-fit component of the description length criterion. Hence, both MMI and ML (since they are equivalent in this case) are special cases of the MDL procedure wherein the complexity term is a constant and is left out of the optimization (all models are assumed to be equally likely and likelihood alone is used as the decision criterion).

## 4.2   Complexity Penalization for Multi-layer Perceptron Models

It has been proposed in the past (Barron, 1989) to use a penalty term of $(k/2) \log N$, where $k$ is the number of parameters (weights and biases) in the network. The origins of this complexity measure lie in general arguments originally proposed by Rissanen (1984). However this penalty term is too large. Cybenko (1990) has

pointed out that existing successful applications of networks have far more parameters than could possibly be justified by a statistical analysis, given the amount of training data used to construct the network. The critical factor lies in the *precision* to which these parameters are stated in the final model. In essence the principle of MDL (and Bayesian techniques) dictates that the data only justifies the stating of any parameter in the model to some finite precision, inversely proportional to the inherent variance of the estimate. Approximate techniques for the calculation of the complexity terms in this manner have been proposed (Weigend, Huberman and Rumelhart, this volume) but a complete description length analysis has not yet appeared in the literature.

### 4.3    Complexity Penalization for a Discrete Network Model

It turns out that there are alternatives to multi-layer perceptrons whose complexity is much easier to calculate. We will look in particular at the rule-based network of Goodman et al. (1990). In this model the hidden units correspond to Boolean combinations of discrete input variables. The link weights from hidden to output (class) nodes are proportional to log conditional probabilities of the class given the activation of a hidden node. The output nodes form estimates of the posterior class probabilities by a simple summation followed by a normalization. The implicit assumption of conditional independence is ameliorated in practice by the fact that the hidden units are chosen in a manner to ensure that the assumption is violated as little as possible.

The complexity penalty for the network is calculated as being $(1/2)\log N$ per link from the hidden to output layers, plus an appropriate coding term for the specification of the hidden units. Hence, the description length of a network with $k$ hidden units would be

$$L = -\sum_{i=1}^{N} \log(\hat{p}(y_i|\underline{x_i})) + k/2 \log N - \sum_{i=1}^{k} \log \pi(o_i)$$

where $o_i$ is the order of the $i$th hidden node and $\pi(o_i)$ is a prior probability on the orders. Using this definition of description length we get from our earlier results on admissible models that the number of hidden units in the architecture is upper bounded by

$$k \leq \frac{NH(C)}{0.5 \log N + \log K + 1}$$

where $K$ is the number of binary input attributes.

### 4.4    Application to a Medical Diagnosis Problem

We consider the application of our techniques to the discovery of a parsimonious network for breast cancer diagnosis, using the discrete network model. A common technique in breast cancer diagnosis is to obtain a fine needle aspirate (FNA) from the patient. The FNA sample is then evaluated under a microscope by a physician who makes a diagnosis. Ground truth in the form of binary class labels ("benign" or "malignant") is obtained by re-examination or biopsy at a later stage. Wolberg and Mangasarian (1991) described the collection of a database of such information.

The feature information consisted of subjective evaluations of nine FNA sample characteristics such as uniformity of cell size, marginal adhesion and mitoses. The training data consists of 439 such FNA samples obtained from real patients which were later assigned class labels. Given that the prior class entropy is almost 1 bit, one can immediately state from our bounds that networks with more than 51 hidden units are inadmissible. Furthermore, as we evaluate different models we can narrow the region of admissibility using the results stated earlier. Figure 1 gives a graphical interpretation of this procedure.

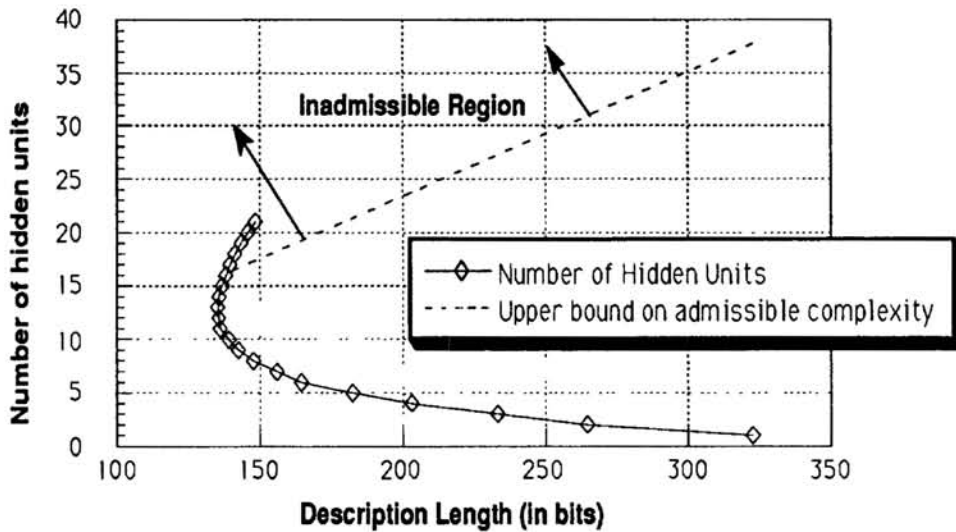

Figure 1. Inadmissible region as a function of description length

The algorithm effectively moves up the left-hand axis, adding hidden units in a greedy manner. Initially the description length (the lower curve) decreases rapidly as we capture the gross structure in the data. For each model that we calculate a description length, we can in turn calculate an upper bound on admissibility (the upper curve) — this bound is linear in description length. Hence, for example by the time we have 5 hidden units we know that any models with more than 21 hidden units are inadmissible. Finally a local minimum of the description length function is reached at 12 units, at which point we know that the optimal solution can have at most 16 hidden units. As matter of interest, the resulting network with 12 hidden units correctly classified 94 of 96 independent test cases.

## 5   Conclusion

There are a variety of related issues which arise in this context which we can only briefly mention due to space constraints. For example, how does the prior "model entropy", $H(\Omega_N) = -\sum_i p(M_i) \log(p(M_i))$, affect the complexity of the search problem? Questions also naturally arise as to how $\Omega_N$ should grow as a function of $N$ in an *incremental* learning scenario.

In conclusion, it should not be construed from this paper that consideration of admissible models is the major factor in inductive inference — certainly the choice of description lengths for the various models and the use of efficient optimization

techniques for seeking the parameters of each model remain the cornerstones of success. Nonetheless, our results provide useful theoretical insight and are practical to the extent that they provide a "sanity check" for model selection in MDL.

## Acknowledgments

The research described in this paper was performed at the Jet Propulsion Laboratories, California Institute of Technology, under a contract with the National Aeronautics and Space Administration. In addition this work was supported in part by the Air Force Office of Scientific Research under grant number AFOSR–90–0199.

## References

A. R. Barron (1989), 'Statistical properties of artificial neural networks,' in *Proceedings of 1989 IEEE Conference on Decision and Control.*

A. R. Barron and T. M. Cover (1991), 'Minimum complexity density estimation,' to appear in *IEEE Trans. Inform. Theory.*

J. Bridle (1990), 'Training stochastic model recognition algorithms as networks can lead to maximum mutual information estimation of parameters,' in D. S. Touretzky (ed.), *Advances in Neural Information Processing Systems 1*, pp.211–217, San Mateo, CA: Morgan Kaufmann.

G. Cybenko (1990), 'Complexity theory of neural networks and classification problems,' preprint.

H. Gish (1991), 'Maximum likelihood training of neural networks,' to appear in *Proceedings of the Third International Workshop on AI and Statistics*, (D. Hand, ed.), Chapman and Hall: London.

R. M. Goodman, C. Higgins, J. W. Miller, and P. Smyth (1990), 'A rule-based approach to neural network classifiers,' in *Proceedings of the 1990 International Neural Network Conference*, Paris, France.

V. A. Kovalevsky (1980), *Image Pattern Recognition*, translated from Russian by A. Brown, New York: Springer Verlag, p.79.

J. Rissanen (1984), 'Universal coding, information, prediction, and estimation,' *IEEE Trans. Inform. Theory*, vol.30, pp.629–636.

P. Smyth (1991), 'Admissible stochastic complexity models for classification problems,' to appear in *Proceedings of the Third International Workshop on AI and Statistics*, (D. Hand, ed.), Chapman and Hall: London.

C. S. Wallace and P. R. Freeman (1987), 'Estimation and inference by compact coding,' *J. Royal Stat. Soc. B*, vol.49, no.3, pp.240–251.

W. H. Wolberg and O. L. Mangasarian (1991), Multi-surface method of pattern separation applied to breast cytology diagnosis, *Proceedings of the National Academy of Sciences*, in press.